# CoFi<sup>Rank</sup>
# Maximum Margin Matrix Factorization for Collaborative Ranking

**Markus Weimer**[*]    **Alexandros Karatzoglou**[†]    **Quoc Viet Le**[‡]    **Alex Smola**[§]

## Abstract

In this paper, we consider collaborative filtering as a ranking problem. We present a method which uses Maximum Margin Matrix Factorization and optimizes ranking instead of rating. We employ structured output prediction to optimize directly for ranking scores. Experimental results show that our method gives very good ranking scores and scales well on collaborative filtering tasks.

## 1 Introduction

Collaborative filtering has gained much attention in the machine learning community due to the need for it in webshops such as those of Amazon, Apple and Netflix. Webshops typically offer personalized recommendations to their customers. The quality of these suggestions is crucial to the overall success of a webshop. However, suggesting the right items is a highly nontrivial task: (1) There are many items to choose from. (2) Customers only consider very few (typically in the order of ten) recommendations. Collaborative filtering addresses this problem by learning the suggestion function for a user from ratings provided by this and other users on items offered in the webshop. Those ratings are typically collected on a five star ordinal scale within the webshops.

Learning the suggestion function can be considered either a rating (classification) or a ranking problem. In the context of rating, one predicts the actual rating for an item that a customer has not rated yet. On the other hand, for ranking, one predicts a preference ordering over the yet unrated items. Given the limited size of the suggestion shown to the customer, both (rating and ranking) are used to compile a top-N list of recommendations. This list is the direct outcome of a ranking algorithm, and can be computed from the results of a rating algorithm by sorting the items according to their predicted rating. We argue that rating algorithms solve the wrong problem, and one that is actually harder: The absolute value of the rating for an item is highly biased for different users, while the ranking is far less prone to this problem.

One approach is to solve the rating problem using regression. For example for the Netflix prize which uses root mean squared error as an evaluation criterion,[1] the most straightforward approach is to use regression. However, the same arguments discussed above apply to regression. Thus, we present an algorithm that solves the ranking problem directly, without first computing the rating.

For collaborative rating, Maximum Margin Matrix Factorization (MMMF) [11, 12, 10] has proven to be an effective means of estimating the rating function. MMMF takes advantage of the collaborative effects: rating patterns from other users are used to estimate ratings for the current user. One key

---

[*]Telecooperation Group, TU Darmstadt, Germany, mweimer@tk.informatik.tu-darmstadt.de
[†]Department of Statistics, TU Wien, alexis@ci.tuwien.ac.at
[‡]Computer Science Department, Stanford University, Stanford, CA 94305, quoc.le@stanford.edu
[§]SML, NICTA, Northbourne Av. 218, Canberra 2601, ACT, Australia, alex.smola@nicta.com.au

[1]We conjecture that this is the case in order to keep the rules simple, since ranking scores are somewhat nontrivial to define, and there are many different ways to evaluate a ranking, as we will see in the following.

advantage of this approach is that it works without feature extraction. Feature extraction is domain specific, e.g. the procedures developed for movies cannot be applied to books. Thus, it is hard to come up with a consistent feature set in applications with many different types of items, as for example at Amazon. Our algorithm is based on this idea of MMMF, but optimizes *ranking* measures instead of rating measures.

Given that only the top ranked items will actually be presented to the user, it is much more important to rank the first items right than the last ones. In other words, it is more important to predict what a user *likes* than what she *dislikes*. In more technical terms, the value of the error for estimation is not uniform over the ratings. All of above reasonings lead to the following goals:

- The algorithm needs to be able to optimize ranking scores directly.
- The algorithm needs to be adaptable to different scores.
- The algorithm should not require any features besides the actual ratings.
- The algorithm needs to scale well and parallelize such as to deal with millions of ratings arising from thousands of items and users with an acceptable memory footprint.

We achieve these goals by combining (a) recent results in optimization, in particular the application of bundle methods to convex optimization problems [14], (b) techniques for representing functions on matrices, in particular maximum margin matrix factorizations [10, 11, 12] and (c) the application of structured estimation for ranking problems. We describe our algorithm COFI$^{\text{RANK}}$ in terms of optimizing the ranking measure Normalized Discounted Cumulative Gain (NDCG).

## 2 Problem Definition

Assume that we have $m$ items and $u$ users. The ratings are stored in the sparse matrix $Y$ where $Y_{i,j} \in \{0, \ldots, r\}$ is the rating of item $j$ by user $i$ and $r$ is some maximal score. $Y_{i,j}$ is 0 if user $i$ did not rate item $j$. In rating, one estimates the missing values in $Y$ directly while we treat this as a ranking task. Additionally, in NDCG [16], the correct order of higher ranked items is more important than that of lower ranked items:

**Definition 1 (NDCG)** *Denote by $y \in \{1, \ldots, r\}^n$ a vector of ratings and let $\pi$ be a permutation of that vector. $\pi_i$ denotes the position of item $i$ after the permutation. Moreover, let $k \in \mathbb{N}$ be a truncation threshold and $\pi_s$ sorts $y$ in decreasing order. In this case the Discounted Cumulative Gains (DCG@k) score [5] and its normalized variant (NDCG@k) are given by*

$$DCG@k(y, \pi) = \sum_{i=1}^{k} \frac{2^{y_{\pi_i}} - 1}{log(i+2)} \quad and \quad NDCG@k(y, \pi) = \frac{DCG@k(y, \pi)}{DCG@k(y, \pi_s)}$$

DCG@k is maximized for $\pi = \pi_s$. The *truncation* threshold $k$ reflects how many recommendations users are willing to consider. NDCG is a normalized version of DCG so that the score is bounded by $[0, 1]$.

Unlike classification and regression measures, DCG is defined on permutations, not absolute values of the ratings. Departing from traditional pairwise ranking measures [4], DCG is position-dependent: Higher positions have more influence on the score than lower positions. Optimizing DCG has gained much interest in the machine learning and information retrieval (e.g. [2]) communities. However, we present the first effort to optimize this measure for collaborative filtering.

To perform estimation, we need a recipe for obtaining the permutations $\pi$. Since we want our system to be scalable, we need a method which scales not much worse than linearly in the number of the items to be ranked. The avenue we pursue is to estimate a matrix $F \in \mathbb{R}^{m \times u}$ and to use the values $F_{ij}$ for the purpose of ranking the items $j$ for user $i$. Given a matrix $Y$ of known ratings we are now able to define the performance of $F$:

$$R(F, Y) := \sum_{i=1}^{u} \text{NDCG@k}(\Pi^i, Y^i), \tag{1}$$

where $\Pi^i$ is $\mathrm{argsort}(-F^i)$, it sorts $F^i$ in decreasing order.[2] While we would like to maximize $R(F, Y_{\mathrm{test}})$ we only have access to $R(F, Y_{\mathrm{train}})$. Hence, we need to restrict the complexity of $F$ to ensure good performance on the test set when maximizing the score on the training set.

## 3 Structured Estimation for Ranking

However, $R(F, Y)$ is non-convex. In fact, it is piecewise constant and therefore clearly not amenable to any type of smooth optimization. To address this issue we take recourse to structured estimation [13, 15]. Note that the scores decompose into a sum over individual users' scores, hence we only need to show how minimizing $-\mathrm{NDCG}(\pi, y)$ can be replaced by minimizing a convex upper bound on the latter. Summing over the users then provides us with a convex bound for all of the terms.[3] Our conversion works in three steps:

1. Converting $\mathrm{NDCG}(\pi, y)$ into a loss by computing the regret with respect to the optimal permutation $\mathrm{argsort}(-y)$.
2. Denote by $\pi$ a permutation (of the $n$ items a user might want to see) and let $f \in \mathbb{R}^n$ be a estimated rating. We design a mapping $\psi(\pi, f) \to \mathbb{R}$ which is linear in $f$ in such a way that maximizing $\psi(\pi, f)$ with respect to $\pi$ yields $\mathrm{argsort}(f)$.
3. We use the convex upper-bounding technique described by [15] to combine regret and linear map into a convex upper bound which we can minimize efficiently.

**Step 1 (Regret Conversion)** Instead of maximizing $\mathrm{NDCG}(\pi, y)$ we may also minimize

$$\Delta(\pi, y) := 1 - \mathrm{NDCG}(\pi, y). \tag{2}$$

$\Delta(\pi, y)$ is nonnegative and vanishes for $\pi = \pi_s$.

**Step 2 (Linear Mapping)** Key in our reasoning is the use of the Polya-Littlewood-Hardy inequality: For any two vectors $a, b \in \mathbb{R}^n$ their inner product is maximized by sorting $a$ and $b$ in the same order, that is $\langle a, b \rangle \leq \langle \mathrm{sort}(a), \mathrm{sort}(b) \rangle$. This allows us to encode the permuation $\pi = \mathrm{argsort}(f)$ in the following fashion: denote by $c \in \mathbb{R}^n$ a decreasing nonnegative sequence, then the function

$$\psi(\pi, f) := \langle c, f_\pi \rangle \tag{3}$$

is linear in $f$ and maximized with respect to $\pi$ for $\mathrm{argsort}(f)$. Since $c_i$ is decreasing by construction, the Polya-Littlewood-Hardy inequality applies. We found that choosing $c_i = (i + 1)^{-0.25}$ produced good results in our experiments. However, we did not formally optimize this parameter.

**Step 3 (Convex Upper Bound)** We adapt a result of [15] which describes how to find convex upper bounds on nonconvex optimization problems.

**Lemma 2** *Assume that $\psi$ is defined as in (3). Moreover let $\pi^* := \mathrm{argsort}(-f)$ be the ranking induced by $f$. Then the following loss function $l(f, y)$ is convex in $f$ and it satisfies $l(f, y) \geq \Delta(y, \pi^*)$.*

$$l(f, y) := \max_\pi \Big[ \Delta(\pi, y) + \langle c, f_\pi - f \rangle \Big] \tag{4}$$

**Proof** We show convexity first. The argument of the maximization over the permutations $\pi$ is a linear and thus convex function in $f$. Taking the maximum over a set of convex functions is convex itself, which proves the first claim. To see that it is an upper bound, we use the fact that

$$l(f, y) \geq \Delta(\pi^*, y) + \langle c, f_{\pi^*} - f \rangle \geq \Delta(\pi^*, y). \tag{5}$$

The second inequality follows from the fact that $\pi^*$ maximizes $\langle c, f_{\pi^*} \rangle$. ∎

## 4   Maximum Margin Matrix Factorization

**Loss**   The reasoning in the previous section showed us how to replace the ranking score with a convex upper bound on a regret loss. This allows us to replace the problem of maximizing $R(F, Y)$ by that of minimizing a convex function in $F$, namely

$$L(F, Y) := \sum_{i=1}^{u} l(F^i, Y^i) \tag{6}$$

**Matrix Regularization**   Having addressed the problem of non-convexity of the performance score we need to find an efficient way of performing capacity control of $F$, since we only have $L(F, Y_{\text{train}})$ at our disposition, whereas we would like to do well on $L(F, Y_{\text{test}})$. The idea to overcome this problem is by means of a regularizer on $F$, namely the one proposed for Maximum Margin Factorization by Srebro and coworkers[10, 11, 12]. The key idea in their reasoning is to introduce a regularizer on $F$ via

$$\Omega[F] := \frac{1}{2} \min_{M,U} \left[ \operatorname{tr} MM^\top + \operatorname{tr} UU^\top \right] \text{ subject to } UM = F. \tag{7}$$

More specifically, [12] show that the above is a proper norm on $F$. While we could use a semidefinite program as suggested in [11], the latter is intractable for anything but the smallest problems.[4] Instead, we replace $F$ by $UM$ and solve the following problem:

$$\underset{M,U}{\text{minimize}} \, L(UM, Y_{\text{train}}) + \frac{\lambda}{2} \left[ \operatorname{tr} MM^\top + \operatorname{tr} UU^\top \right] \tag{8}$$

Note that the above matrix factorization approach effectively allows us to learn an item matrix $M$ and a user matrix $U$ which will store the specific properties of users and items respectively. This approach *learns* the features of the items and the users. The dimension $d$ of $M \in \mathbb{R}^{d \times m}$ and $U \in \mathbb{R}^{d \times u}$ is chosen mainly based on computational concerns, since a full representation would require $d = \min(m, u)$. On large problems the storage requirements for the user matrix can be enormous and it is convenient to choose $d = 10$ or $d = 100$.

**Algorithm**   While (8) may not be jointly convex in $M$ and $U$ any more, it still is convex in $M$ and $U$ individually, whenever the other term is kept fixed. We use this insight to perform alternating subspace descent as proposed by [10]. Note that the algorithm does *not* guarantee global convergence, which is a small price to pay for computational tractability.

> **repeat**
> For fixed $M$ minimize (8) with respect to $U$.
> For fixed $U$ minimize (8) with respect to $M$.
> **until** No more progress is made or a maximum iteration count has been reached.

Note that on problems of the size of Netflix the matrix $Y$ has $10^8$ entries, which means that the number of iterations is typically time limited. We now discuss a general optimization method for solving regularized convex optimization problems. For more details see [14].

## 5   Optimization

**Bundle Methods**   We discuss the optimization over the user matrix $U$ first, that is, consider the problem of minimizing

$$R(U) := L(UM, Y_{\text{train}}) + \frac{\lambda}{2} \operatorname{tr} UU^\top \tag{9}$$

The regularizer $\operatorname{tr} UU^\top$ is rather simple to compute and minimize. On the other hand, $L$ is expensive to compute, since it involves maximizing $l$ for all users.

Bundle methods, as proposed in [14] aim to overcome this problem by performing successive Taylor approximations of $L$ and by using them as lower bounds. In other words, they exploit the fact that

$$L(UM, Y_{\text{train}}) \geq L(UM', Y_{\text{train}}) + \operatorname{tr}(M - M')^\top \partial_M L(UM', Y) \forall M, M'.$$

**Algorithm 1** Bundle Method($\epsilon$)
---
Initialize $t = 0, U_0 = 0, b_0 = 0$ and $H = \infty$
**repeat**
    Find minimizer $U_t$ and value $L$ of the optimization problem

$$\underset{U}{\text{minimize}} \ \max_{0 \leq j \leq t} \left[ \operatorname{tr} U_j^\top M + b_j \right] + \frac{\lambda}{2} \operatorname{tr} U^\top U.$$

    Compute $U_{t+1} = \partial_U L(U_t M, Y_{\text{train}})$
    Compute $b_{t+1} = L(U_t M, Y_{\text{train}}) - \operatorname{tr} U_{t+1} M_t$
    **if** $H' := \operatorname{tr} U_{t+1}^\top M_t + b_{t+1} + \frac{\lambda}{2} \operatorname{tr} U U^\top \leq H$ **then**
        Update $H \leftarrow H'$
    **end if**
**until** $H - L \leq \epsilon$
---

Since this holds for arbitrary $M'$, we may pick a set of $M_i$ and use the maximum over the Taylor approximations at locations $M_i$ to lower-bound $L$. Subsequently, we minimize this piecewise linear lower bound in combination with $\frac{\lambda}{2} \operatorname{tr} U U^\top$ to obtain a new location where to compute our next Taylor approximation and iterate until convergence is achieved. Algorithm 1 provides further details.

As we proceed with the optimization, we obtain increasingly tight lower bounds on $L(UM, Y_{\text{train}})$. One may show [14] that the algorithm converges to $\epsilon$ precision with respect to the minimizer of $R(U)$ in $O(1/\epsilon)$ steps. Moreover, the initial distance from the optimal solution enters the bound only logarithmically.

After solving the optimization problem in $U$ we switch to optimizing over the item matrix $M$. The algorithm is virtually identical to that in $U$, except that we now need to use the regularizer in $M$ instead of that in $U$. We find experimentally that a small number of iterations (less than 10) is more than sufficient for convergence.

**Computing the Loss** So far we simply used the loss $l(f, y)$ of (4) to define a convex loss without any concern to its computability. To implement Algorithm 1, however, we need to be able to solve the maximization of $l$ with respect to the set of permutations $\pi$ efficiently. One may show that computing the $\pi$ which maximizes $l(f, y)$ is possible by solving the inear assignment problem $min \sum_i \sum_j C_{i,j} X_{i,j}$ with the cost matrix:

$$C_{i,j} = \kappa_i \frac{2^{Y[j]} - 1}{DCG(Y, k, \pi_s) log(i + 1)} - c_i f_j \text{ with } \kappa_i = \begin{cases} 1 & \text{if } i < k, \\ 0 & \text{otherwise} \end{cases}$$

Efficient algorithms [7] based on the Hungarian Marriage algorithm (also referred to as the Kuhn-Munkres algorithm) exist for this problem [8]: it turns out that this integer programming problem can be solved by invoking a linear program. This in turn allows us to compute $l(f, y)$ efficiently.

**Computing the Gradients** The second ingredient needed for applying the bundle method is to compute the gradients of $L(F, Y)$ with respect to $F$, since this allows us to compute gradients with respect to $M$ and $U$ by applying the chain rule:

$$\partial_M L(UM, Y) = U^\top \partial_F L(\mathcal{X}, F, Y) \text{ and } \partial_U L(UM, Y) = \partial_F L(\mathcal{X}, F, Y)^\top M$$

$L$ decomposes into losses on individual users as described in (6). For each user $i$ only row $i$ of $F$ matters. It follows that $\partial_F L(F, Y)$ is composed of the gradients of $l(F^i, Y^i)$. Note that for $l$ defined as in (4) we know that

$$\partial_{F^i} l(F^i, Y^i) = [c - c_{\bar{\pi}^{-1}}].$$

Here we denote by $\bar{\pi}$ the maximizer of of the loss and $c_{\bar{\pi}^{-1}}$ denotes the application of the inverse permutation $\bar{\pi}^{-1}$ to the vector $c$.

# 6 Experiments

We evaluated CoFi$^{\text{RANK}}$ with the NDCG loss just defined (denoted by CoFi$^{\text{RANK}}$-NDCG) as well as with loss functions which optimize ordinal regression (CoFi$^{\text{RANK}}$-Ordinal) and regression (CoFi$^{\text{RANK}}$-Regression). CoFi$^{\text{RANK}}$-Ordinal applies the algorithm described above to preference ranking by optimizing the preference ranking loss. Similarly, CoFi$^{\text{RANK}}$-Regression optimizes for regression using the root mean squared loss. We looked at two real world evaluation settings: "weak" and "strong" [9] generalization on three publicly available data sets: EachMovie, MovieLens and Netflix. Statistics for those can be found in table 1.

| Dataset | Users | Movies | Ratings |
|---------|-------|--------|---------|
| EachMovie | 61265 | 1623 | 2811717 |
| MovieLens | 983 | 1682 | 100000 |
| Netflix | 480189 | 17770 | 100480507 |

*Table 1:* Data set statistics

**Weak generalization**   is evaluated by predicting the rank of unrated items for users known at training time. To do so, we randomly select $N = 10, 20, 50$ ratings for each user for training and and evaluate on the remaining ratings. Users with less then $20, 30, 60$ rated movies where removed to ensure that the we could evaluate on at least 10 movies per user We compare CoFi$^{\text{RANK}}$-NDCG, CoFi$^{\text{RANK}}$-Ordinal, CoFi$^{\text{RANK}}$-Regression and MMMF [10]. Experimental results are shown in table 2.

For all CoFi$^{\text{RANK}}$ experiments, we choose $\lambda = 10$. We did not optimize for this parameter. The results for MMMF were obtained using MATLAB code available from the homepage of the authors of [10]. For those, we used $\lambda = \frac{1}{1.9}$ for EachMovie, and $\lambda = \frac{1}{1.6}$ for MovieLens as it is reported to yield the *best* results for MMMF. In all experiments, we choose the dimensionality of $U$ and $M$ to be 100. All CoFi$^{\text{RANK}}$ experiments and those of MMMF on MovieLens were repeated ten times. Unfortunately, we underestimated the runtime and memory requirements of MMMF on EachMovie. Thus, we cannot report results on this data set using MMMF.

Additionally, we performed some experiments on the Netflix data set. However, we cannot compare to any of the other methods on that data set as to the best of our knowledge, CoFi$^{\text{RANK}}$ is the first collaborative ranking algorithm to be applied to this data set, supposedly because of its large size.

**Strong generalization**   is evaluated on users that were not present at training time. We follow the procedure described in [17]: Movies with less than 50 ratings are discarded. The 100 users with the most rated movies are selected as the test set and the methods are trained on the remaining users. In evaluation, 10, 20 or 50 ratings from those of the 100 test users are selected. For those ratings, the user training procedure is applied to optimize $U$. $M$ is kept fixed in this process to the values obtained during training. The remaining ratings are tested using the same procedure as for the weak

| | Method | N=10 | N=20 | N=50 |
|---|--------|------|------|------|
| **EachMovie** | CoFi$^{\text{RANK}}$-NDCG | $0.6562 \pm 0.0012$ | $0.6644 \pm 0.0024$ | $0.6406 \pm 0.0040$ |
| | CoFi$^{\text{RANK}}$-Ordinal | $\mathbf{0.6727 \pm 0.0309}$ | $\mathbf{0.7240 \pm 0.0018}$ | $\mathbf{0.7214 \pm 0.0076}$ |
| | CoFi$^{\text{RANK}}$-Regression | $0.6114 \pm 0.0217$ | $0.6400 \pm 0.0354$ | $0.5693 \pm 0.0428$ |
| **MovieLens** | CoFi$^{\text{RANK}}$-NDCG | $0.6400 \pm 0.0061$ | $0.6307 \pm 0.0062$ | $0.6076 \pm 0.0077$ |
| | CoFi$^{\text{RANK}}$-Ordinal | $0.6233 \pm 0.0039$ | $0.6686 \pm 0.0058$ | $\mathbf{0.7169 \pm 0.0059}$ |
| | CoFi$^{\text{RANK}}$-Regression | $\mathbf{0.6420 \pm 0.0252}$ | $0.6509 \pm 0.0190$ | $0.6584 \pm 0.0187$ |
| | MMMF | $0.6061 \pm 0.0037$ | $\mathbf{0.6937 \pm 0.0039}$ | $0.6989 \pm 0.0051$ |
| **Netflix** | CoFi$^{\text{RANK}}$-NDCG | $0.6081$ | $0.6204$ | |
| | CoFi$^{\text{RANK}}$-Regression | $0.6082$ | $0.6287$ | |

*Table 2:* Results for the weak generalization setting experiments. We report the NDCG@10 accuracy for various numbers of training ratings used per user. For most results we report the mean over ten runs and the standard deviation. We also report the p-values for the best vs. second best score.

| | Method | N=10 | N=20 | N=50 |
|---|---|---|---|---|
| **EachMovie** | CoFi$^{\text{RANK}}$-NDCG | **0.6367 ± 0.001** | **0.6619 ± 0.0022** | **0.6771 ± 0.0019** |
| | GPR | 0.4558 ± 0.015 | 0.4849 ± 0.0066 | 0.5375 ± 0.0089 |
| | CGPR | 0.5734 ± 0.014 | 0.5989 ± 0.0118 | 0.6341 ± 0.0114 |
| | GPOR | 0.3692 ± 0.002 | 0.3678 ± 0.0030 | 0.3663 ± 0.0024 |
| | CGPOR | 0.3789 ± 0.011 | 0.3781 ± 0.0056 | 0.3774 ± 0.0041 |
| | MMMF | 0.4746 ± 0.034 | 0.4786 ± 0.0139 | 0.5478 ± 0.0211 |
| | | | | |
| **MovieLens** | CoFi$^{\text{RANK}}$-NDCG | **0.6237 ± 0.0241** | **0.6711 ± 0.0065** | 0.6455 ± 0.0103 |
| | GPR | 0.4937 ± 0.0108 | 0.5020 ± 0.0089 | 0.5088 ± 0.0141 |
| | CGPR | 0.5101 ± 0.0081 | 0.5249 ± 0.0073 | 0.5438 ± 0.0063 |
| | GPOR | 0.4988 ± 0.0035 | 0.5004 ± 0.0046 | 0.5011 ± 0.0051 |
| | CGPOR | 0.5053 ± 0.0047 | 0.5089 ± 0.0044 | 0.5049 ± 0.0035 |
| | MMMF | 0.5521 ± 0.0183 | 0.6133 ± 0.0180 | **0.6651 ± 0.0190** |

*Table 3:* The NGDC@10 accuracy over ten runs and the standard deviation for the strong generalization evaluation.

generalization. We repeat the whole process 10 times and again use $\lambda = 10$ and a dimensionality of 100. We compare CoFi$^{\text{RANK}}$-NDCG to Gaussian Process Ordinal Regression (GPOR) [3] Gaussian Process Regression (GPR) and the collaborative extensions (CPR, CGPOR) [17]. Table 3 shows our results compared to the ones from [17].

CoFi$^{\text{RANK}}$ performs strongly compared to most of the other tested methods. Particularly in the strong generalization setting CoFi$^{\text{RANK}}$ outperforms the existing methods in almost all the settings. Note that all methods except CoFi$^{\text{RANK}}$ and MMMF use additional extracted features which are either provided with the dataset or extracted from the IMDB. MMMF and CoFi$^{\text{RANK}}$ only rely on the rating matrix. In the weak generalization experiments on the MovieLens data, CoFi$^{\text{RANK}}$ performs better for $N = 20$ but is marginally outperformed by MMMF for the $N = 10$ and $N = 50$ cases. We believe that with proper parameter tuning, CoFi$^{\text{RANK}}$ will perform better in these cases.

## 7 Discussion and Summary

CoFi$^{\text{RANK}}$ is a novel approach to collaborative filtering which solves the ranking problem faced by webshops directly. It can do so faster and at a higher accuracy than approaches which learn a *rating* to produce a *ranking*. CoFi$^{\text{RANK}}$ is adaptable to different loss functions such as NDCG, Regression and Ordinal Regression in a plug-and-play manner. Additionally, CoFi$^{\text{RANK}}$ is well suited for privacy concerned applications, as the optimization itself does not need ratings from the users, but only gradients.

Our results, which we obtained *without parameters tuning*, are on par or outperform several of the most successful approaches to collaborative filtering like MMMF, even when they are used with *tuned parameters*. CoFi$^{\text{RANK}}$ performs best on data sets of realistic sizes such as EachMovie and significantly outperforms other approaches in the strong generalization setting.

In our experiments, CoFi$^{\text{RANK}}$ shows to be very fast. For example, training on EachMovie with $N = 10$ can be done in less than ten minutes and uses less than $80MB$ of memory on a laptop. For $N = 20$, CoFi$^{\text{RANK}}$ obtained a NDCG@10 of 0.72 after the first iteration, which also took less than ten minutes. This is the highest NDCG@10 score on that data set we are aware of (apart from the result of CoFi$^{\text{RANK}}$ after convergence). A comparison to MMMF in that regard is difficult, as it is implemented in MATLAB and CoFi$^{\text{RANK}}$ in C++. However, CoFi$^{\text{RANK}}$ is more than ten times faster than MMMF while using far less memory. In the future, we will exploit the fact that the algorithm is easily parallelizable to obtain even better performance on current multi-core hardware as well as computer clusters. Even the current implementation allows us to report the first results on the Netflix data set for direct ranking optimization.

**Acknowledgments:** Markus Weimer is funded by the German Research Foundation as part of the Research Training Group 1223: "Feedback-Based Quality Management in eLearning".

**Software:** CoFi$^{\text{RANK}}$ is available from http://www.cofirank.org

## Footnotes

[2] $M^i$ denotes row $i$ of matrix $M$. Matrices are written in upper case, while vectors are written in lower case.

[3] This also opens the possibility for parallelization in the implementation of the algorithm.

[4]In this case we optimize over $\begin{bmatrix} A & F \\ F^\top & B \end{bmatrix} \succeq 0$ where $\Omega[F]$ is replaced by $\frac{1}{2}[\operatorname{tr} A + \operatorname{tr} B]$.

# References

[2] C. J. Burges, Q. V. Le, and R. Ragno. Learning to rank with nonsmooth cost functions. In B. Schölkopf, J. Platt, and T. Hofmann, editors, *Advances in Neural Information Processing Systems 19*, 2007.

[3] W. Chu and Z. Ghahramani. Gaussian processes for ordinal regression. *J. Mach. Learn. Res.*, 6:1019–1041, 2005.

[4] R. Herbrich, T. Graepel, and K. Obermayer. Large margin rank boundaries for ordinal regression. In A. J. Smola, P. L. Bartlett, B. Schölkopf, and D. Schuurmans, editors, *Advances in Large Margin Classifiers*, pages 115–132, Cambridge, MA, 2000. MIT Press.

[5] K. Jarvelin and J. Kekalainen. IR evaluation methods for retrieving highly relevant documents. In *ACM Special Interest Group in Information Retrieval (SIGIR)*, pages 41–48. New York: ACM, 2002.

[7] R. Jonker and A. Volgenant. A shortest augmenting path algorithm for dense and sparse linear assignment problems. *Computing*, 38:325–340, 1987.

[8] H.W. Kuhn. The Hungarian method for the assignment problem. *Naval Research Logistics Quarterly*, 2:83–97, 1955.

[9] B. Marlin. Collaborative filtering: A machine learning perspective. Masters thesis, University of Toronto, 2004.

[10] J. Rennie and N. Srebro. Fast maximum margin matrix factoriazation for collaborative prediction. In *Proc. Intl. Conf. Machine Learning*, 2005.

[11] N. Srebro, J. Rennie, and T. Jaakkola. Maximum-margin matrix factorization. In L. K. Saul, Y. Weiss, and L. Bottou, editors, *Advances in Neural Information Processing Systems 17*, Cambridge, MA, 2005. MIT Press.

[12] N. Srebro and A. Shraibman. Rank, trace-norm and max-norm. In P. Auer and R. Meir, editors, *Proc. Annual Conf. Computational Learning Theory*, number 3559 in Lecture Notes in Artificial Intelligence, pages 545–560. Springer-Verlag, June 2005.

[13] B. Taskar, C. Guestrin, and D. Koller. Max-margin Markov networks. In S. Thrun, L. Saul, and B. Schölkopf, editors, *Advances in Neural Information Processing Systems 16*, pages 25–32, Cambridge, MA, 2004. MIT Press.

[14] C.H. Teo, Q. Le, A.J. Smola, and S.V.N. Vishwanathan. A scalable modular convex solver for regularized risk minimization. In *Conference on Knowledge Discovery and Data Mining*, 2007.

[15] I. Tsochantaridis, T. Joachims, T. Hofmann, and Y. Altun. Large margin methods for structured and interdependent output variables. *J. Mach. Learn. Res.*, 6:1453–1484, 2005.

[16] E. Voorhees. Overview of the TREC 2001 question answering track. In *Text REtrieval Conference (TREC) Proceedings*. Department of Commerce, National Institute of Standards and Technology, 2001. NIST Special Publication 500-250: The Tenth Text REtrieval Conference (TREC 2001).

[17] S. Yu, K. Yu, V. Tresp, and H. P. Kriegel. Collaborative ordinal regression. In W.W. Cohen and A. Moore, editors, *Proc. Intl. Conf. Machine Learning*, pages 1089–1096. ACM, 2006.

